# Stochastic Relational Models
# for Discriminative Link Prediction

**Kai Yu**
NEC Laboratories America
Cupertino, CA 95014

**Wei Chu**
CCLS, Columbia University
New York, NY 10115

**Shipeng Yu, Volker Tresp, Zhao Xu**
Siemens AG, Corporate Research & Technology,
81739 Munich, Germany

## Abstract

We introduce a Gaussian process (GP) framework, *stochastic relational models* (SRM), for learning social, physical, and other relational phenomena where interactions between entities are observed. The key idea is to model the stochastic structure of entity relationships (i.e., links) via a tensor interaction of multiple GPs, each defined on one type of entities. These models in fact define a set of nonparametric priors on infinite dimensional tensor matrices, where each element represents a relationship between a tuple of entities. By maximizing the marginalized likelihood, information is exchanged between the participating GPs through the entire relational network, so that the dependency structure of links is messaged to the dependency of entities, reflected by the adapted GP kernels. The framework offers a discriminative approach to link prediction, namely, predicting the existences, strengths, or types of relationships based on the partially observed linkage network as well as the attributes of entities (if given). We discuss properties and variants of SRM and derive an efficient learning algorithm. Very encouraging experimental results are achieved on a toy problem and a user-movie preference link prediction task. In the end we discuss extensions of SRM to general relational learning tasks.

## 1   Introduction

Relational learning concerns the modeling of physical, social, or other phenomena, where rich types of entities interact via complex relational structures. If compared to the traditional machine learning settings, the entity relationships provide additional structural information.

A simple example of a relational setting is the user-movie rating database, which contains user entities with user attributes (e.g., age, gender, education), movie entities with movie attributes (e.g., year, genre, director), and ratings (i.e., relations between users and movies). A typical relational learning problem is *entity classification*, for example, segmenting users into groups. One may apply usual clustering or classification methods based on a *flat structure* of data, where each user is associated with a vector of user attributes. However a sound model should additionally explore the user-movie relations as well as even the movie attributes, since like-minded users tend to give similar ratings on the same movie, and may like (or dislike) movies with similar genres. Relational learning addresses this and similar situation where it is not natural to transform the data into a flat structure.

Entity classification in a relational setting has gained considerable attentions, like webpage classification using both textual contents and hyperlinks. However, in other occasions relationships themselves are often of central interest. For example, one may want to predict protein-protein in-

teractions, or in another application, user ratings for products. The family of these problems has been called *link prediction*[1], which is the primary topic of this paper. In general, link prediction includes link existence prediction (i.e., does a link exist?), link classification (i.e., what type of the relationship?), and link regression (i.e., how does the user rate the item?).

In this paper we propose a family of *stochastic relational models* (SRM) for link prediction and other relational learning tasks. The key idea of SRM is a stochastic link-wise process induced by a tensor interplay of multiple entity-wise Gaussian processes (GP). These models in fact define a set of nonparametric priors on an infinite dimensional tensor matrix, where each element represents a relationship between a tuple of entities. By maximizing the marginalized likelihood, information is exchanged between the participating GPs through the entire relational network, so that the dependency structure of links is messaged to the dependency of entities, reflected by the learned entity-wise GP kernels (i.e., GP covariance functions). SRM is *discriminative* because training is on a conditional model of links. We present various models of SRM and address the computational issue, which is crucial in link prediction because the number of potential links grows exponentially with the entity size. SRM has shown encouraging results in our experiments.

This paper is organized as follows. We introduce the stochastic relational models in Sec. 2, and describe the algorithms for inference and parameter estimation in Sec. 3 and Sec. 4, followed by Sec. 5 on implementation details. Then we discuss the related work in Sec. 6 and report experimental results in Sec. 7, followed by conclusions and extensions in Sec. 8.

## 2 Stochastic Relational Models

We first consider pairwise asymmetric links $r$ between entities $u \in \mathcal{U}$ and $v \in \mathcal{V}$. The two sets $\mathcal{U}$ and $\mathcal{V}$ can be identical or different. We use $u$ or $v$ to represent the attribute vectors of entities or their identity when entity attributes are unavailable. Note that $r_{i,n} \equiv r(u_i, v_n)$ does not have to be identical to $r_{n,i}$ when $\mathcal{U} = \mathcal{V}$, i.e. relationships can be asymmetrical. Extensions to involve more than two entity sets, multi-way relations (i.e., links connecting more than two entities), and symmetric links are straightforward and will be briefly discussed in Sec. 8.

We assume that the observable links $r$ are derived as local measurements of a real-valued *latent relational function* $t : \mathcal{U} \times \mathcal{V} \to \mathbb{R}$, and each link $r_{i,n}$ is solely dependent on its latent value $t_{i,n}$, modeled by the likelihood $p(r_{i,n}|t_{i,n})$. The focus of this paper is a family of *stochastic relational processes* acting on $\mathcal{U} \times \mathcal{V}$, the space of entity pairs, to generate the latent relational function $t$, via a tensor interaction of two independent entity-specific GPs, one acting on $\mathcal{U}$ and the other on $\mathcal{V}$. We call them *processes* because $\mathcal{U}$ and $\mathcal{V}$ can both encompass an infinite number of entities. Let the relational processes be characterized by *hyperparameters* $\theta = \{\theta_\Sigma, \theta_\Omega\}$, $\theta_\Sigma$ for the GP kernel function on $\mathcal{U}$ and $\theta_\Omega$ for the GP kernel function on $\mathcal{V}$, a SRM thus defines a Bayesian prior $p(t|\theta)$ for the latent variables $t$. Let $\mathbb{I}$ be the index set of entity pairs having observed links, the marginal likelihood (also called evidence) under such a prior is

$$p(\mathbf{R}_\mathbb{I}|\theta) = \int \prod_{(i,n)\in\mathbb{I}} p(r_{i,n}|t_{i,n})p(t|\theta)dt, \quad \theta = \{\theta_\Sigma, \theta_\Omega\} \tag{1}$$

where $\mathbf{R}_\mathbb{I} = \{r_{i,n}\}_{(i,n)\in\mathbb{I}}$. We estimate the hyperparameters $\theta$ by maximizing the evidence, which is an empirical Bayesian approach to learning the *relational structure* of data. Once $\theta$ are determined, we can predict the link for a new pair of entities via marginalization over the *a posteriori* $p(t|\mathbf{R}_\mathbb{I}, \theta)$.

### 2.1 Choices for the Piror $p(t|\theta)$

In order to represent a rich class of link structures, $p(t|\theta)$ should be sufficiently expressive. In the following subsections, we will present three cases of $p(t|\theta)$, from specific to general, by gradually extending conventional GP models.

### 2.1.1 A Brief Introduction to Gaussian Processes

A GP defines a nonparametric prior distribution over functions in Bayesian inference. A random real-valued function $f : \mathcal{X} \to \mathbb{R}$ follows a GP prior, denoted by $\mathcal{GP}(\mu, \Sigma)$, if for *every* finite set

$\{x_i\}_{i=1}^N$, $\mathbf{f} = \{f(x_i)\}_{i=1}^N$ follows a multivariate Gaussian distribution with mean $\boldsymbol{\mu} = \{\mu(x_i)\}_{i=1}^N$ and covariance (or kernel) $\boldsymbol{\Sigma} = \{\Sigma(x_i, x_j; \theta_\Sigma)\}_{i,j=1}^N$ with parameter $\theta_\Sigma$. Given $\mathcal{D} = \{x_i, y_i\}_{i=1}^N$, where $y_i$ is a measurement of $f(x_i)$ corrupted by Gaussian noise, one can make predictions via the marginal likelihood $p(y|x, \mathcal{D}, \theta_\Sigma) = \int p(y|f, x) p(f|\mathcal{D}, \theta_\Sigma) df$. For non-Gaussian measurement processes, as in classification models, the integral cannot be solved analytically, and approximation for inference is required. A comprehensive coverage of GP models can be found in [9].

### 2.1.2 Hierarchical Gaussian Processes

By observing the relational data collectively, one may notice that two entities $u_i$ and $u_j$ in $\mathcal{U}$ demonstrate correlated relationships to entities in $\mathcal{V}$. For example, two users often show opposite or close opinions on movies, or two hub web pages are co-linked by a set of other authority web pages. In this case, the dependency structure of links can be reduced to a dependency structure of entities in $\mathcal{U}$. A hierarchical GP (HGP) model [13], originally proposed for *multi-task learning*, can be conveniently applied in such a situation. The model assumes that, for every $v \in \mathcal{V}$, its relational function $t(\cdot, v) : \mathcal{U} \to \mathbb{R}$ is an i.i.d. sample drawn from a common entity-wise GP with covariance $\Sigma : \mathcal{U} \times \mathcal{U} \to \mathbb{R}$. This provides a case of $p(t|\theta)$ in a SRM, where $\theta = \theta_\Sigma$ determines the GP kernel function $\Sigma$. Optimizing the GP kernel $\Sigma$ via evidence maximization means to learn the dependency of entities in $\mathcal{U}$, summarized over all the entities $v \in \mathcal{V}$.

There is a drawback if applying HGP to link prediction. The model only learns a one-side structure, while ignoring the dependency in $\mathcal{V}$. In particular, the attributes of entities $v$ cannot be incorporated even if their entity attributes are available. However, for the mentioned applications, it also makes sense to explore the dependency between movies, or the dependency between authority web pages.

### 2.1.3 Tensor Gaussian Processes

To overcome the shortcoming of HGP, we consider a more complex SRM, which employs two GP kernel functions $\Sigma : \mathcal{U} \times \mathcal{U} \to \mathbb{R}$ and $\Omega : \mathcal{V} \times \mathcal{V} \to \mathbb{R}$. The model explains the relational dependency through the entity dependencies of both $\mathcal{V}$ and $\mathcal{U}$. Let $\theta = \{\theta_\Sigma, \theta_\Omega\}$, we describe a stochastic relational process $p(t|\theta)$ as the following:

**Definition 2.1** (Tensor Gaussian Processes). *Given two sets $\mathcal{U}$ and $\mathcal{V}$, a collection of random variables $\{t(u,v)|t : \mathcal{U} \times \mathcal{V} \to \mathbb{R}\}$ follow a tensor Gaussian processes (TGP), if for every finite sets $\{u_1, \ldots, u_N\}$ and $\{v_1, \ldots, v_M\}$, random variables $\mathbf{T} = \{t(u_i, v_n)\}$, organized into an $N \times M$ matrix, have a matrix-variate normal distribution*

$$\mathcal{N}_{N \times M}(\mathbf{T}|\mathbf{B}, \boldsymbol{\Sigma}, \boldsymbol{\Omega}) = (2\pi)^{-\frac{MN}{2}} |\boldsymbol{\Sigma}|^{-\frac{M}{2}} |\boldsymbol{\Omega}|^{-\frac{N}{2}} \text{etr}\left\{-\frac{1}{2}\boldsymbol{\Sigma}^{-1}(\mathbf{T}-\mathbf{B})\boldsymbol{\Omega}^{-1}(\mathbf{T}-\mathbf{B})^\top\right\}$$

*characterized by mean $\mathbf{B} = \{b(u_i, v_n)\}$ and positive definite covariance matrices $\boldsymbol{\Sigma} = \{\Sigma(u_i, u_j; \theta_\Sigma)\}$ and $\boldsymbol{\Omega} = \{\Omega(v_n, v_m; \theta_\Omega)\}$. This random process is denoted as $\mathcal{TGP}(b, \Sigma, \Omega)$.[2]*

In the above theorem $\text{etr}[\cdot]$ is a shortcut for $\exp[\text{trace}(\cdot)]$. It is easy to see that the model reduces to the HGP model if $\boldsymbol{\Omega} = \mathbf{I}$. As a key difference, the new model treats the relational function $t$ as a *whole* sample from a TGP, instead of being formed by i.i.d. functions in the HGP model. Let $\text{vec}(\mathbf{T}^\top) = [t_{1,1}, t_{1,2}, \ldots, t_{1,M}, t_{2,1}, \ldots, t_{2,M}, \ldots, t_{N,M}]^\top$. If $\mathbf{T} \sim \mathcal{N}_{N \times M}(\mathbf{T}|\mathbf{B}, \boldsymbol{\Sigma}, \boldsymbol{\Omega})$, then $\text{vec}(\mathbf{T}^\top) \sim \mathcal{N}(0, \boldsymbol{\Upsilon})$, where the covariance $\boldsymbol{\Upsilon} = \boldsymbol{\Sigma} \otimes \boldsymbol{\Omega}$ is the Kronecker product of two covariance matrices [6]. In other words, TGP is in fact a GP for the relational function $t$, where the kernel function $\Upsilon : (\mathcal{U} \times \mathcal{V}) \times (\mathcal{U} \times \mathcal{V}) \to \mathbb{R}$ is defined via a tensor product of two GP kernels $\text{Cov}(t_{i,n}, t_{j,m}) = \Upsilon[(u_i, v_n), (u_j, v_m)] = \Sigma(u_i, u_j)\Omega(v_n, v_m)$. The model explains the dependence structure of links by the dependence structure of participating entities.

It is well known that a linear predictive model has a GP interpretation if its linear weights follow a Gaussian prior. A similar connection exists for TGP.

**Theorem 2.2.** *Let $\mathcal{U} \subseteq \mathbb{R}^P$, $\mathcal{V} \subseteq \mathbb{R}^Q$, and $\mathbf{W} \sim \mathcal{N}_{P \times Q}(0, \mathbf{I}_P, \mathbf{I}_Q)$, where $\mathbf{I}_P$ denotes a $P \times P$ identity matrix and $\langle \cdot, \cdot \rangle$ denotes the inner product, then the bilinear function $t(u, v) = u^\top \mathbf{W} v$ follows $\mathcal{TGP}(0, \Sigma, \Omega)$ with $\Sigma(u_i, u_j) = \langle u_i, u_j \rangle$ and $\Omega(v_n, v_m) = \langle v_n, v_m \rangle$.*

The proof is straightforward through $\text{Cov}[t(u_i, v_n), t(u_j, v_m)] = \langle u_i, u_j \rangle \langle v_n, v_m \rangle$ and $\text{E}[t(u_i, v_n)] = 0$, where $\text{E}[\cdot]$ means expectation. In practice, the linear model will help to provide an efficient discriminative approach to link prediction.

It appears that link prediction using TGP is almost the same as a normal GP regression or classification, except that the hyperparameters $\theta$ now have two parts, $\theta_\Sigma$ for $\Sigma$ and $\theta_\Omega$ for $\Omega$. Unfortunately TGP inference suffers from a serious computational problem – it does not scale well for even a small size of entities. For example, if there is a fixed portion of missing data for pairwise relationships between $N$ $u$-entities and $M$ $v$-entities, the size of observations scales in $\mathcal{O}(NM)$. Since GP inference has the complexity cubic to the size of data, the complexity $\mathcal{O}(N^3 M^3)$ of TGP is computationally prohibitive.

### 2.1.4 A Family of Stochastic Processes for Entity Relationships

To improve the scalability of SRM, and also describe the relational dependency in a way similar to TGP, we propose a family of stochastic processes $p(t|\theta)$ for entity relationships.

**Definition 2.3** (Stochastic Relational Processes). *A relational function $t : \mathcal{U} \times \mathcal{V} \to \mathbb{R}$ is said to follow a stochastic relational process (SRP), if $t(u, v) = \frac{1}{\sqrt{d}} \sum_{k=1}^{d} f_k(u) g_k(v)$, $f_k(u) \overset{iid}{\sim} \mathcal{GP}(0, \Sigma)$, $g_k(v) \overset{iid}{\sim} \mathcal{GP}(0, \Omega)$. We denote $t \sim \mathcal{SRP}_d(0, \Sigma, \Omega)$, where $d$ is the degrees of freedom.*

Interestingly, there exists an intimate connection between SRP and TGP:

**Theorem 2.4.** $\mathcal{SRP}_d(0, \Sigma, \Omega)$ *converges to* $\mathcal{TGP}(0, \Sigma, \Omega)$ *in the limit* $d \to \infty$.

*Proof.* Based on the central limit theory, for every $(u_i, v_n)$, $t_{i,n} \equiv t(u_i, v_n)$ converges to a Gaussian random variable. In the next steps, we prove $\text{E}[t_{i,n}] = 0$ and $\text{Cov}(t_{i,n}, t_{j,m}) = \text{E}[t_{i,n} t_{j,m}] = \frac{1}{d} \{ \sum_{k=1}^{d} \text{E}[f_k(u_i) f_k(u_j) g_k(v_n) g_k(v_m)] + \sum_{k \neq \kappa} \text{E}[f_k(u_i) f_\kappa(u_j) g_k(v_n) g_\kappa(v_m)] \} = \frac{1}{d} \sum_{k=1}^{d} \text{E}[f_k(u_i) f_k(u_j) g_k(v_n) g_k(v_m)] = \Sigma(u_i, u_j) \Omega(v_n, v_m)$. $\square$

The theorem suggests that there is a constructive definition of TGP, where relationships are formed via interactions between infinite samples from two GPs. Moreover, given a sufficiently large $d$, SRP will provide a close approximation to TGP.

SRP is a general family of priors for modeling the relationships between entities, in which HGP and TGP are special cases. The generalization offers several advantages: (1) SRP can model symmetric links between the same set of entities. If we build a generative process where $\mathcal{U} = \mathcal{V}$, $\Sigma = \Omega$ and $f_k = g_k$, then on every finite sets $\{u_i\}_{i=1}^{N}$, $\mathbf{T} = \{t(u_i, u_j)\}$ is always a symmetric matrix; (2) Given a fixed $d$, the inference with SRP obtains a much reduced complexity. In Sec. 3 we will introduce an inference algorithm that scales in $\mathcal{O}[d(N^3 + M^3)]$, which is much less than $\mathcal{O}(N^3 M^3)$.

## 2.2 Choices for the Likelihood $p(r_{i,n}|t_{i,n})$

The likelihood term describes the dependency of observable relationships on the latent variables. It should be tailored to the types of observations to be modeled. Here we list three possible situations:

• **Binary Classification**: Relationships may take categorical states, e.g., "cue" or "no cue" in disease-treatment relationship prediction. It is popular to consider binary classification and employ the probit function to model the Bernoulli distribution over class labels, i.e. $p(r_{i,n}|t_{i,n}) = \Phi(r_{i,n} t_{i,n})$, where $\Phi(\cdot)$ is a cumulative normal function, and $r_{i,n} \in \{-1, +1\}$.

• **Regression**: In this case we consider $r_{i,n}$ taking continuous values. For example, one may want to predict the rating of user $u$ for movie $v$. The corresponding likelihood function is essentially defined by a noise model, e.g. a univariate Gaussian noise with variance $\rho^2$ and zero mean.

• **One-class Problem**: Sometimes one observed the presence of links between entities, for example, the hyperlinks between web pages. Based on the *open-world assumption*, if a web page does not link to another, it does not mean that they are unrelated. Therefore, we employ the likelihood $p(r_{i,n}|t_{i,n}) = \Phi(r_{i,n} t_{i,n} - b)$ for those observed links $r_{i,n} = 1$, where $b$ is an offset.

# 3 Inference with Laplacian Approximation

We have described the relational model under a prior of SRP, in which HGP and TGP are subcases. Now we develop the inference algorithm to compute the sufficient statistics of the *a posteriori* distribution of latent variables.

Let $\mathbf{F} = \{f_{i,k}\}$, $\mathbf{G} = \{g_{n,k}\}$, $\mathbf{f}_k = [f_{1,k}, \ldots, f_{N,k}]^\top$ and $\mathbf{g}_k = [g_{1,k}, \ldots, g_{M,k}]^\top$, where $f_{i,k} = f_k(u_i)$, $g_{n,k} = g_k(v_n)$. Then the posterior distribution $p(\mathbf{F}, \mathbf{G}|\mathbf{R}_\mathbb{I}, \theta)$ is proportional to the joint distribution of the *complete data*:

$$p\left(\mathbf{R}_\mathbb{I}, \mathbf{F}, \mathbf{G}|\theta\right) \propto \prod_{(i,n)\in\mathbb{I}} p\left(r_{i,n} \Big| \frac{\sum_{k=1}^d f_{i,k} g_{n,k}}{\sqrt{d}}\right) \exp\left\{-\frac{1}{2}\sum_{k=1}^d \left[\mathbf{f}_k^\top \mathbf{\Sigma}^{-1}\mathbf{f}_k + \mathbf{g}_k^\top \mathbf{\Omega}^{-1}\mathbf{g}_k\right]\right\}$$

An exact inference is intractable due to the coupling between $f_{i,k}$ and $g_{n,k}$ in the likelihood term. In this paper we apply Laplacian approximation, which approximates $p(\mathbf{F}, \mathbf{G}|\mathbf{R}_\mathbb{I}, \theta)$ by a multivariate normal distribution $q(\mathbf{F}, \mathbf{G}|\boldsymbol{\beta})$ with sufficient statistics $\boldsymbol{\beta}$. At the first step, we compute the means by finding the mode in the posterior,

$$\{\mathbf{F}^*, \mathbf{G}^*\} = \arg\min_{\{\mathbf{F},\mathbf{G}\}} \left\{J(\mathbf{F}, \mathbf{G}) = -\log p(\mathbf{R}_\mathbb{I}, \mathbf{F}, \mathbf{G}|\theta)\right\} \tag{2}$$

We solve the minimization by the conjugate gradient method. The gradients are calculated by

$$\frac{\partial J(\mathbf{F}, \mathbf{G})}{\partial \mathbf{F}} = \frac{1}{\sqrt{d}}\mathbf{S}\mathbf{G} + \mathbf{\Sigma}^{-1}\mathbf{F}, \qquad \frac{\partial J(\mathbf{F}, \mathbf{G})}{\partial \mathbf{G}} = \frac{1}{\sqrt{d}}\mathbf{S}^\top \mathbf{F} + \mathbf{\Omega}^{-1}\mathbf{G},$$

where $\mathbf{S} \in \mathbb{R}^{N\times M}$ have elements $s_{i,n} = \partial[-\log p(r_{i,n}|t_{i,n})]/\partial t_{i,n}$, $t_{i,n} = \sum_{k=1}^d f_{i,k}g_{n,k}/\sqrt{d}$, if $(i,n) \in \mathbb{I}$, otherwise $s_{i,n} = 0$. At the second step we calculate the covariance by $\mathbf{C} = \mathbf{H}^{-1}$, where $\mathbf{H}$ is the Hessian, i.e., the second-order derivatives of $J(\mathbf{F}, \mathbf{G})$ with respect to $\{\mathbf{F}, \mathbf{G}\}$. However the inverse is prohibitive because $\mathbf{H}$ is a huge matrix. To reduce the complexity, we assume that there exist disjoint groups of latent variables, each group is second-order independent to any other at their equilibriums. We factorize the approximating distribution as $q(\mathbf{F}, \mathbf{G}|\boldsymbol{\beta}) = \prod_{k=1}^d q(\mathbf{f}_k|\mathbf{f}_k^*, \mathbf{\Sigma}_k)q(\mathbf{g}_k|\mathbf{g}_k^*, \mathbf{\Omega}_k)$, where $\mathbf{f}_k^*$ and $\mathbf{g}_k^*$ are the solution from Eq. (2), and $\mathbf{\Sigma}_k, \mathbf{\Omega}_k$ are the covariances matrices. This follows the facts: (1) Each $\mathbf{f}_k$ (or $\mathbf{g}_k$) indirectly interacts with other $\mathbf{f}_\kappa$ (or $\mathbf{g}_\kappa$), $\kappa \neq k$, through the sum $\sum_{\kappa\neq k} \mathbf{f}_\kappa \mathbf{g}_\kappa^\top$, indicating that $\mathbf{f}_k$ (or $\mathbf{g}_k$) across different $k$ are only loosely dependent to each other, especially for a large $d$; (2) The dependency between $f_{i,k}$ and $g_{n,k}$ is witnessed via at most only one observation in $\mathbf{R}_\mathbb{I}$. Therefore we can compute the Hessian for each group separately and obtain the covariances:

$$\mathbf{\Sigma}_k = (\mathbf{\Phi}^{(k)} + \mathbf{\Sigma}^{-1})^{-1}, \quad \text{with} \quad \phi_{i,i}^{(k)} = \sum_{n:(i,n)\in\mathbb{I}} \frac{\zeta_{i,n}g_{n,k}^2}{d}, \quad \phi_{i,j}^{(k)} = 0 \tag{3}$$

$$\mathbf{\Omega}_k = (\mathbf{\Psi}^{(k)} + \mathbf{\Omega}^{-1})^{-1}, \quad \text{with} \quad \psi_{n,n}^{(k)} = \sum_{i:(i,n)\in\mathbb{I}} \frac{\zeta_{i,n}f_{i,k}^2}{d}, \quad \psi_{n,m}^{(k)} = 0 \tag{4}$$

where $\zeta_{i,n} = \partial^2[-\log p(r_{i,n}|t_{i,n})]/\partial t_{i,n}^2$. Then we obtain the sufficient statistics $\mathbf{F}^*$, $\mathbf{G}^*$, $\{\mathbf{\Sigma}_k\}$ and $\{\mathbf{\Omega}_k\}$. Finally we note that, the posterior distribution of $\{\mathbf{F}, \mathbf{G}\}$ has many modes (Simply, shuffling the order of latent dimensions or changing the signs of both $\mathbf{f}_k$ and $\mathbf{g}_k$ does not change the probability.). However each mode is equally well in constructing the relational function $t$.

# 4 Structural Learning by Hyperparameter Estimation

We assign a hyper prior $p(\theta|\boldsymbol{\alpha})$ and estimate $\theta$ by maximizing a penalized marginal likelihood

$$\theta^* = \arg\max_{\theta=\{\theta_\Sigma, \theta_\Omega\}} \left\{\log \int_\mathbf{G}\int_\mathbf{F} p(\mathbf{R}_\mathbb{I}, \mathbf{F}, \mathbf{G}|\theta)d\mathbf{F}d\mathbf{G} + \log p(\theta|\boldsymbol{\alpha})\right\} \tag{5}$$

So far the optimization (5) is quite general. In principal, it allows to learn some parametric forms of kernel functions $\Sigma(u_i, u_j; \theta_\Sigma)$ and $\Omega(v_n, v_m; \theta_\Omega)$ that are generalizable to new entities. In this

paper we particularly consider an situation where entity attributes are not fully informative or even absent. Therefore we introduce a direct parameterization $\boldsymbol{\theta}_\Sigma = \boldsymbol{\Sigma}$, $\theta_\Omega = \boldsymbol{\Omega}$, and assign conjugate inverse-Wishart priors $\boldsymbol{\Sigma} \sim \mathcal{IW}_N(\lambda d, \boldsymbol{\Sigma}_0)$ and $\boldsymbol{\Omega} \sim \mathcal{IW}_M(\lambda d, \boldsymbol{\Omega}_0)$, namely

$$p(\boldsymbol{\Sigma}|\lambda d, \boldsymbol{\Sigma}_0) \propto \det(\boldsymbol{\Sigma})^{-\frac{\lambda d}{2}} \operatorname{etr}\left(-\frac{\lambda d}{2}\boldsymbol{\Sigma}^{-1}\boldsymbol{\Sigma}_0\right),$$

$$p(\boldsymbol{\Omega}|\lambda d, \boldsymbol{\Omega}_0) \propto \det(\boldsymbol{\Omega})^{-\frac{\lambda d}{2}} \operatorname{etr}\left(-\frac{\lambda d}{2}\boldsymbol{\Omega}^{-1}\boldsymbol{\Omega}_0\right),$$

where $\lambda > 0$ so that $\lambda d$ denotes the degrees of freedom, $\boldsymbol{\Sigma}_0$ and $\boldsymbol{\Omega}_0$ are the base kernels. Then we apply an iterative expectation-maximization (EM) algorithm to solve the problem (5). In the E-step, we follow Sec. 3 to compute $q(\mathbf{F}, \mathbf{G}|\boldsymbol{\beta})$. In the M-step, we update the hyperparameters by maximizing the expected log-likelihood of the *complete data*

$$\max_{\{\boldsymbol{\Sigma}, \boldsymbol{\Omega}\}} \mathrm{E}_q\left[\log p(\mathbf{R}_\mathbb{I}, \mathbf{F}, \mathbf{G}|\boldsymbol{\Sigma}, \boldsymbol{\Omega})\right] + \log p(\boldsymbol{\Sigma}|\lambda d, \boldsymbol{\Sigma}_0) + \log p(\boldsymbol{\Omega}|\lambda d, \boldsymbol{\Omega}_0)$$

where $\mathrm{E}_q[\cdot]$ is the expectation over $q(\mathbf{F}, \mathbf{G}|\boldsymbol{\beta})$. Due to the conjugacy of the hyper prior, the maximization have an analytical solution,

$$\boldsymbol{\Sigma} = \frac{\lambda\boldsymbol{\Sigma}_0 + \frac{1}{d}\sum_{k=1}^d(\mathbf{f}_k^*\mathbf{f}_k^{*\top} + \boldsymbol{\Sigma}_k)}{\lambda + 1}, \quad \boldsymbol{\Omega} = \frac{\lambda\boldsymbol{\Omega}_0 + \frac{1}{d}\sum_{k=1}^d(\mathbf{g}_k^*\mathbf{g}_k^{*\top} + \boldsymbol{\Omega}_k)}{\lambda + 1}. \tag{6}$$

## 5 Implementation Details

The parameters $\boldsymbol{\Sigma}_0, \boldsymbol{\Omega}_0, d$ and $\lambda$ have to be pre-specified. We let the base kernels have the form $\Sigma_0(u_i, u_j) = (1 - a)\gamma(u_i, u_j) + a\delta_{i,j}$ and $\Omega_0(v_n, v_m) = (1 - \eta)\xi(v_n, v_m) + \eta\delta_{n,m}$, where $1 \geq a, \eta \geq 0$, $\delta$ is a Dirac delta kernel ($\delta_{i,j} = 1$ if $i = j$, otherwise $\delta_{i,j} = 0$), $\gamma(\cdot, \cdot)$ and $\xi(\cdot, \cdot)$ are some kernel functions defined on entity attributes, which reflect our prior notion of similarities between entities. We use $a$ and $\eta$ to penalize the effects of $\gamma(\cdot, \cdot)$ and $\xi(\cdot, \cdot)$, respectively, when entity attributes are deficient. If the attributes are unavailable, we set $a = \eta = 1$. The dimensionality $d$ should be properly chosen, otherwise a too small $d$ may deteriorate the modeling flexibility. We determine $d$ and $\lambda$ based on the prediction performance on a validation set of links. The learning algorithm iterates the E-step with Eq. (2), (3), (4), and the M-step with Eq. (6) until convergence. In the experiments of this paper we use $p(r_{i,n}|t_{i,n}^*)$ to make predictions, where $t^*$ is computed from $\mathbf{F}^*$ and $\mathbf{G}^*$. In a longer version the predictive uncertainty of $t_{i,n}$ will be considered.

## 6 Related Work

There is a history of probabilistic relational models (PRM) [8] in machine learning. Getoor et al. [5] introduced link uncertainty and defined a generative model for both entity attributes and links. Recently, [12] and [7] independently introduced an infinite (hidden) relational model to avoid the difficulty of structural learning in PRM by explaining links via a potentially infinite number of hidden states of entities. Since discriminatively trained models generally outperform generative models in prediction tasks, Taskar et al. proposed relational Markov networks (RMNs) for link prediction [11], by describing a conditional distribution of links given entity attributes and other links. RMN has to define a class of potential functions on cliques of random variables based on the observed relational structure. Compared to RMN, SRM is nonparametric because structural information (e.g., cliques as well as the classes of potential functions) is not pre-defined but learned from data. Very recently a GP model was developed to learn from undirected graphs [4], which turns out to be a special rank-one case of SRM with $d = 1$, $\Sigma = \Omega$, and $f_k = h_k$. In another work [1] a SVM using a tensor kernel based on user and item attributes was used to predict user ratings on items, which is similar to our TGP case and suffers a salability problem. When attributes are deficient or unavailable, the model does not work well, while SRM can learn informative kernels purely from only links (see Sec. 7). SRM is interestingly related to the recent fast maximum-margin matrix factorization (MMMF) in [10]. If we fix $\Sigma$ and $\Omega$ as uninformative Dirac kernels, the mode of our Laplacian approximation is equivalent to the solution of Eq.(5) in [10] with the loss function $l(r_{i,n}, t_{i,n}) = -\log p(r_{i,n}|t_{i,n})$. However SRM significantly differs from MMMF in two important aspects: (1) SRM is a supervised predictive model because entity attributes enter the model by forming informative priors $(\Sigma, \Omega)$ and hyper priors $(\Sigma_0, \Omega_0)$; (2) More importantly, SRM deals with

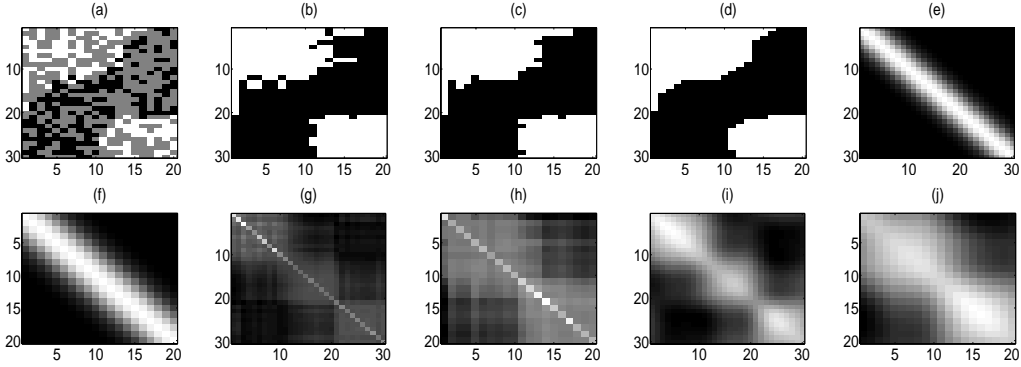

Figure 1: Link prediction on synthetic data: (a) training data, where black entry means positive links, white means negative links, and gray means missing; (b) prediction of MMMF (classification rate 0.906); (c) prediction of SRM with noninformative prior (classification rate 0.942); (d) prediction of SRM with informative prior (classification rate 0.965); (e-f) informative $\Sigma_0$ and $\Omega_0$; (g-h) learned $\Sigma$ and $\Omega$ with noninformative prior; (i-j) learned $\Sigma$ and $\Omega$ with informative prior.

structural learning by adapting the kernels and marginalizing out the latent relational function, while MMMF only estimates the mode of the latent relational function with fixed Dirac kernels.

## 7  Experiments

**Synthetic Data**: We generated two sets of entities $\mathcal{U} = \{u_i\}_{i=1}^{20}$ and $\mathcal{V} = \{v_n\}_{n=1}^{30}$ on a real line such that $u_i = 0.1i$ and $v_n = 0.1n$. The positions of entities were used to compute two RBF kernels that serve as informative $\Sigma_0$ and $\Omega_0$. Then we further made a deformation on the real line to form 2 clusters in $\mathcal{U}$ and 3 clusters in $\mathcal{V}$. RBF function computed on the deformed scale gives two kernel matrices $\Sigma$ and $\Omega$ whose diagonal block structure reflects the clusters. Binary links between $\mathcal{U}$ and $\mathcal{V}$ are obtained by taking the sign of a function, which is a sample from $\mathcal{TGP}(0, \Sigma, \Omega)$. We randomly withdrew $50\%$ of links for training, and left the remaining for test (see Fig. 1-(a)). We performed two variants of SRM, one with informative $\Sigma_0$ and $\Omega_0$ (see Fig. 1-(e,f)) and the other with noninformative Dirac kernels $\Sigma_0 = \Omega_0 = \mathbf{I}$, and compared with MMMF [10]. In all the cases we set $d = 20$. The classification accuracy rates of two SRMs, $0.942$ and $0.965$, are both better than $0.906$ of MMMF. As shown in Fig. 1, the block structures of learned kernels indicate that both SRMs can learn the cluster structure of entities from links. The structural kernel optimization enables SRM to outperform MMMF, even with a completely noninformative prior. Note that the informative prior really helps SRM to achieve the best accuracy.

**Eachmovie Data**: We tested our algorithms on a data set from [3], which is a subset of EachMovie data, containing 5000 users' ratings, i.e., $1, 2, 3, 4, 5$, or $6$, on 1623 movies. We selected the first 1000 users and organized the data into a $1000 \times 1623$ table with $63,592$ observed ratings. We compared SRM with MMMF in a regression task to predict the 'rating link' between users and movies. In SRM we set $\Sigma_0 = \Omega_0 = \mathbf{I}$. For both methods the dimensionality was chosen as $d = 20$. In MMMF we used the square error loss. We repeated the experiments for 10 times, where at each time we randomly withdrew $70\%$ ratings for training and left the remaining for test. Root-mean-square error (RMSE) and mean-absolute error (MAE) were used to evaluate the accuracy. The results of all the repeats, as well as the means and standard deviations, are shown in Table 1 and Table 2. Compared to MMMF, SRM significantly reduces the prediction error by over $12\%$ in terms of both RMSE and MAE.

## 8  Conclusions and Future Extensions

In this paper we proposed a stochastic relational model (SRM) for learning relational data. Entity relationships are modeled by a tensor interaction of multiple Gaussian processes (GPs). We proposed a family of relational processes and showed its convergence to a tensor Gaussian process if the degrees of freedom goes to infinity. The process imposes an effective prior on the entity relationships,

Table 1: User-movie rating prediction error measured by RMSE

| Repeats | 1 | 2 | 3 | 4 | 5 | 6 | 7 | 8 | 9 | 10 | mean ± std. |
|---|---|---|---|---|---|---|---|---|---|---|---|
| MMMF | 1.366 | 1.367 | 1.372 | 1.377 | 1.363 | 1.368 | 1.356 | 1.380 | 1.358 | 1.373 | 1.368 ± 0.008 |
| SRM | 1.195 | 1.199 | 1.192 | 1.200 | 1.198 | 1.209 | 1.204 | 1.208 | 1.189 | 1.209 | **1.200±0.007** |

Table 2: User-movie rating prediction error measured by MAE

| Repeats | 1 | 2 | 3 | 4 | 5 | 6 | 7 | 8 | 9 | 10 | mean ± std. |
|---|---|---|---|---|---|---|---|---|---|---|---|
| MMMF | 1.067 | 1.066 | 1.074 | 1.076 | 1.066 | 1.073 | 1.060 | 1.074 | 1.062 | 1.072 | 1.060±0.006 |
| SRM | 0.924 | 0.928 | 0.924 | 0.923 | 0.924 | 0.934 | 0.931 | 0.932 | 0.918 | 0.933 | **0.927± 0.005** |

and leads to a discriminative link prediction model. We demonstrated the excellent results of SRM on a synthetic data set and a user-movie rating prediction problem.

Though the current work focused on the application of link prediction, the model can be used for general relational learning purposes. There are several directions to extend the current model: (1) SRM can describe a joint distribution of entity links and entity classes conditioned on entity-wise GP kernels. Therefore entity classification can be solved in a relational context; (2) One can extend SRM to model multi-way relations where more than two entities participate in a single relationship; (3) SRM can also be applied to model pairwise relations between multiple entity sets, where kernel updates amount to propagation of information through the entire relational network; (4) As discussed in Sec. 2.1.2, SRM is a natural extension of hierarchical Bayesian multi-task models, by explicitly modeling the dependency over tasks. In a recent work [2] a tensor GP for multi-task learning was independently suggested; (5) Finally, it is extremely important to make the algorithm scalable to very large relational data, like the Netflix problem, containing about 480,000 users and 17,000 movies.

## Acknowledgement

The authors thank Andreas Krause, Chris Williams, Shenghuo Zhu, and Wei Xu for the fruitful discussions.

## Footnotes

[1]We will use "link" and "relationship" interchangeably throughout this paper.

[2]Hereafter we always assume $b(u, v) = 0$ in TGP for simplicity.

## References

[1] J. Basilico and T. Hofmann. Unifying collaborative and content-based filtering. In *Proceedings of the 21st International Conference on Machine Learning (ICML)*, 2004.

[2] E. V. Bonilla, F. V. Agakov, and C. K. I. Williams. Kernel multi-task learning using task-specific features. In *Proceedings of the 11th International Conference on Artificial Intelligence and Statistics (AISTATS)*, 2007. To appear.

[3] J. S. Breese, D. Heckerman, and C. Kadie. Empirical analysis of predictive algorithms for collaborative filtering. In *Proceedings of the 14th Conference on Uncertainty in Artificial Intelligence (UAI)*, 1998.

[4] W. Chu, V. Sindhwani, Z. Ghahramani, and S. S. Keerthi. Relational learning with gaussian processes. In *Neural Information Processing Systems (NIPS)*, 2007. To appear.

[5] L. Getoor, E. Segal, B. Taskar, and D. Koller. Probabilistic models of text and link structure for hypertext classification. In *Proceedings ICJAI Workshop on Text Learning: Beyond Supervision*, 2001.

[6] Arjun K. Gupta and Daya K. Naga. *Matrix Variate Distributions*. 1999.

[7] C. Kemp, J. B. Tenenbaum, T. L. Griffiths, T. Yamada, and N. Ueda. Learning systems of concepts with an infinite relational model. In *Proceedings of the 21st National Conference on Artificial Intelligence (AAAI)*, 2006.

[8] D. Koller and A. Pfeffer. Probabilistic frame-based systems. In *Proceedings of National Conference on Artificial Intelligence (AAAI)*, 1998.

[9] C. Rasmussen and C. K. I. Williams. *Gaussian Processes for Machine Learning*. MIT Press, 2006.

[10] Jason D. M. Rennie and Nati Srebro. Fast maximum margin matrix factorization for collaborative prediction. In *Proceedings of the 22nd International Conference on Machine Learning (ICML)*, 2005.

[11] B. Taskar, M. F. Wong, P. Abbeel, and D. Koller. Link prediction in relational data. In *Neural Information Processing Systems Conference (NIPS)*, 2004.

[12] Z. Xu, V. Tresp, K. Yu, and H.-P. Kriegel. Infinite hidden relational models. In *Proceedings of the 22nd International Conference on Uncertainty in Artificial Intelligence (UAI)*, 2006.

[13] K. Yu, V. Tresp, and A. Schwaighofer. Learning Gaussian processes from multiple tasks. In *Proceedings of 22nd International Conference on Machine Learning (ICML)*, 2005.
